# Instance-Based Relevance Feedback for Image Retrieval

**Giorgio Giacinto and Fabio Roli**
Department of Electrical and Electronic Engineering
University of Cagliari
Piazza D'Armi, Cagliari – Italy 09121
*{giacinto,roli}@diee.unica.it*

## Abstract

High retrieval precision in content-based image retrieval can be attained by adopting relevance feedback mechanisms. These mechanisms require that the user judges the quality of the results of the query by marking all the retrieved images as being either relevant or not. Then, the search engine exploits this information to adapt the search to better meet user's needs. At present, the vast majority of proposed relevance feedback mechanisms are formulated in terms of search model that has to be optimized. Such an optimization involves the modification of some search parameters so that the nearest neighbor of the query vector contains the largest number of relevant images. In this paper, a different approach to relevance feedback is proposed. After the user provides the first feedback, following retrievals are not based on k-nn search, but on the computation of a relevance score for each image of the database. This score is computed as a function of two distances, namely the distance from the nearest non-relevant image and the distance from the nearest relevant one. Images are then ranked according to this score and the top $k$ images are displayed. Reported results on three image data sets show that the proposed mechanism outperforms other state-of-the-art relevance feedback mechanisms.

## 1  Introduction

A large number of content-based image retrieval (CBIR) systems rely on the vector representation of images in a multidimensional feature space representing low-level image characteristics, e.g., color, texture, shape, etc. [1]. Content-based queries are often expressed by visual examples in order to retrieve from the database the images that are "similar" to the examples. This kind of retrieval is often referred to as *K* nearest-neighbor retrieval.  It is easy to see that the effectiveness of content-based image retrieval systems (CBIR) strongly depends on the choice of the set of visual features, on the choice of the "metric" used to model the user's perception of image similarity, and on the choice of the image used to query the database [1]. Typically, if we allow different users to mark the images retrieved with a given query as

relevant or non-relevant, different subsets of images will be marked as relevant. Accordingly, the need for mechanisms to adapt the CBIR system response based on some feedback from the user is widely recognized.

It is interesting to note that while *relevance feedback* mechanisms have been first introduced in the information retrieval field [2], they are receiving more attention in the CBIR field (Huang). The vast majority of relevance feedback techniques proposed in the literature is based on modifying the values of the search parameters as to better represent the *concept* the user bears in mind. To this end, search parameters are computed as a function of the relevance values assigned by the user to *all* the images retrieved so far. As an example, relevance feedback is often formulated in terms of the modification of the query vector, and/or in terms of adaptive similarity metrics. [3]-[7]. Recently, pattern classification paradigms such as SVMs have been proposed [8]. Feedback is thus used to model the *concept* of relevant images and adjust the search consequently.

Concept modeling may be difficult on account of the distribution of relevant images in the selected feature space. "Narrow domain" image databases allows extracting good features, so that images bearing similar concepts belong to compact clusters. On the other hand, "broad domain" databases, such as image collection used by graphic professionals, or those made up of images from the Internet, are more difficult to subdivide in cluster because of the high variability of concepts [1]. In these cases, it is worth extracting only low level, non-specialized features, and image retrieval is better formulated in terms of a search problem rather then concept modeling.

The present paper aims at offering an original contribution in this direction. Rather then modeling the concept of "relevance" the user bears in mind, feedback is used to assign each image of the database a *relevance* score. Such a score depends only from two dissimilarities (distances) computed against the images already marked by the user: the dissimilarity from the set of relevant images, and the dissimilarity from the set of non-relevant images. Despite its computational simplicity, this mechanism allows outperforming state-of-the-art relevance feedback mechanisms both on "narrow domain" databases, and on "broad domain" databases.

This paper is organized as follows. Section 2 illustrates the idea behind the proposed mechanism and provides the basic assumptions. Section 3 details the proposed relevance feedback mechanism. Results on three image data sets are presented in Section 4, where performances of other relevance feedback mechanisms are compared. Conclusions are drawn in Section 5.

## 2   Instance-based relevance estimation

The proposed mechanism has been inspired by classification techniques based on the "nearest case" [9]-[10]. Nearest-case theory provided the mechanism to compute the dissimilarity of each image from the sets of relevant and non–relevant images. The ratio between the nearest relevant image and the nearest non-relevant image has been used to compute the degree of relevance of each image of the database [11]. The present section illustrates the rationale behind the use of the nearest-case paradigm.

Let us assume that each image of the database has been represented by a number of low-level features, and that a (dis)similarity measure has been defined so that the proximity between pairs of images represents some kind of "conceptual" similarity. In other words, the chosen feature space and similarity metric is meaningful at least for a restricted number of users.

A search in image databases is usually performed by retrieving the $k$ most similar images with respect to a given query. The dimension of $k$ is usually small, to avoid displaying a large number of images at a time. Typical values for $k$ are between 10 and 20. However, as the "relevant" images that the user wishes to retrieve may not fit perfectly with the similarity metric designed for the search engine, the user may be interested in *exploring* other regions of the feature space. To this end, the user marks the subset of "relevant" images out of the $k$ retrieved. Usually, such *relevance feedback* is used to perform a new $k$-nn search by modifying some search parameters, i.e., the position of the query point, the similarity metric, and other tuning parameters [1]-[7]. Recent works proposed the use of support vector machine to learn the distribution of relevant images [8]. These techniques require some assumption about the general form of the distribution of relevant images in the feature space. As it is difficult to make any assumption about such a distribution for broad domain databases, we propose to exploit the information about the relevance of the images retrieved so far in a nearest-neighbor fashion.

Nearest-neighbor techniques, as used in statistical pattern recognition, case-based reasoning, or instance-based learning, are effective in all applications where it is difficult to produce a high-level generalization of a "class" of objects [9]-[10],[12]-[13]. Relevance learning in content base image retrieval may well fit into this definition, as it is difficult to provide a general model that can be adapted to represent different concepts of similarity. In addition, the number of available cases may be too small to estimate the optimal set of parameters for such a general model. On the other hand, it can be more effective to use each "relevant" image as well as each "non-relevant" image, as "cases" or "instances" against which the images of the database should be compared. Consequently, we assume that an image is as much as relevant as much as its dissimilarity from the *nearest relevant* image is small. Analogously, an image is as much as non-relevant as much as its dissimilarity from the *nearest non-relevant* image is small.

## 3   Relevance Score Computation

According to previous section, each image of the database can be thus characterized by a "degree of relevance" and a "degree of non-relevance" according to the dissimilarities from the nearest relevant image, and from the nearest non-relevant image, respectively. However, it should be noted that these degrees should be treated differently because only "relevant" images represent a "concept" in the user's mind, while "non-relevant" images may represent a number of other concepts different from user's interest. In other words, while it is meaningful to treat the degree of relevance as a degree of membership to the *class* of relevant images, the same does not apply to the degree of non-relevance. For this reason, we propose to use the "degree of non-relevance" to weight the "degree of relevance".

Let us denote with $R$ the subset of indexes $j \in \{1,...,k\}$ related to the set of relevant images retrieved so far and the original query (that is relevant by default), and with $NR$ the subset of indexes $j \in (1,...,k\}$ related to the set of non-relevant images retrieved so far. For each image $\mathbf{I}$ of the database, according to the nearest neighbor rule, let us compute the dissimilarity from the nearest image in $R$ and the dissimilarity from the nearest image in $NR$. Let us denote these dissimilarities as $dR(\mathbf{I})$ and $dNR(\mathbf{I})$, respectively. The value of $dR(\mathbf{I})$ can be clearly used to measure the degree of relevance of image $\mathbf{I}$, assuming that small values of $dR(\mathbf{I})$ are related to very relevant images. On the other hand, the hypothesis that image $\mathbf{I}$ is relevant to the user's query can be supported by a high value of $dNR(\mathbf{I})$. Accordingly, we defined the relevance score

$$relevance(\mathbf{I}) = \left(1 + \frac{dR(\mathbf{I})}{dN(\mathbf{I})}\right)^{-1} \tag{1}$$

This formulation of the score can be easily explained in terms of a distance-weighted 2-nn estimation of the posterior probability that image **I** is relevant. The 2 nearest neighbors are made up of the nearest relevant image, and the nearest non-relevant image, while the weights are computed as the inverse of the distance from the nearest neighbors.

The relevance score computed according to equation (1) is then used to rank the images and the first $k$ are presented to the user.

## 4   Experimental results

In order to test the proposed method and compare it with other methods described in the literature, three image databases have been used: the MIT database, a database contained in the UCI repository, and a subset of the Corel database. These databases are currently used for assessing and comparing relevance feedback techniques [5],[7],[14].

The MIT database was collected by the MIT Media Lab (ftp://whitechapel.media.mit.edu/pub/VisTex). This database contains 40 texture images that have been manually classified into fifteen classes. Each of these images has been subdivided into sixteen non-overlapping images, obtaining a data set with 640 images. Sixteen Gabor filters were used to characterise these images, so that each image is represented by a 16-dimensional feature vector [14].

The database extracted from the UCI repository (http://www.cs.uci.edu/mlearn/MLRepository.html) consists of 2,310 outdoor images. The images are subdivided into seven data classes (brickface, sky, foliage, cement, window, path, and grass). Nineteen colour and spatial features characterise each image. (Details are reported in the UCI web site).

The database extracted from the Corel collection is available at the KDD-UCI repository (http://kdd.ics.uci.edu/databases/CorelFeatures/CorelFeatures.data.html). We used a subset made up of 19513 images, manually subdivided into 43 classes. For each image, four sets of features were available at the web site. In this paper, we report the results related to the Color Moments (9 features), and the Co-occurrence Texture (16 features) feature sets

For each dataset, the Euclidean distance metric has been used. A linear normalisation procedure has been performed, so that each feature takes values in the range between 0 and 1.

For the first two databases, each image is used as a query, while for the Corel database, 500 images have been randomly extracted and used as query, so that all the 43 classes are represented. At each retrieval iteration, twenty images are returned. Relevance feedback is performed by marking images belonging to the same class of the query as relevant, and all other images as non-relevant. The user's query itself is included in the set of relevant images. This experimental set up affords an objective comparison among different methods, and is currently used by many researchers [5],[7],[14]. Results are evaluated in term of the retrieval precision averaged over all the considered queries. The precision is measured as the fraction of relevant images contained in the 20 top retrieved images.

As the first two databases are of the "narrow domain" type, while the third is of the "broad domain" type, this experimental set-up allowed a thorough testing of the proposed technique.

For the sake of comparison, retrieval performances obtained with two methods recently described in the literature are also reported: MindReader [3] which modifies the query vector and the similarity metric on account of features relevance, and Bayes QS (Bayesian Query Shifting) which is based on query reformulation [7]. These two methods have been selected because they can be easily implemented, and their performances can be compared to those provided by a large number of relevance feedback techniques proposed in the CBIR literature (see for example results presented in [15]). It is worth noting that results presented in different papers cannot be directly compared to each other because they are not related to a common experimental set-up. However, as they are related to the same data sets with *similar* experimental set-up, a qualitative comparisons let us conclude that the performance of the two above techniques are quite close to other results in the literature.

## 4.1    Experiments with the MIT database

This database can be considered of the "narrow domain" type as it contains only images of textures of 40 different types. In addition, the selected feature space is very suited to measure texture similarity.

Figure 1 show the performances of the proposed relevance feedback mechanism and those of the two techniques used for comparison.

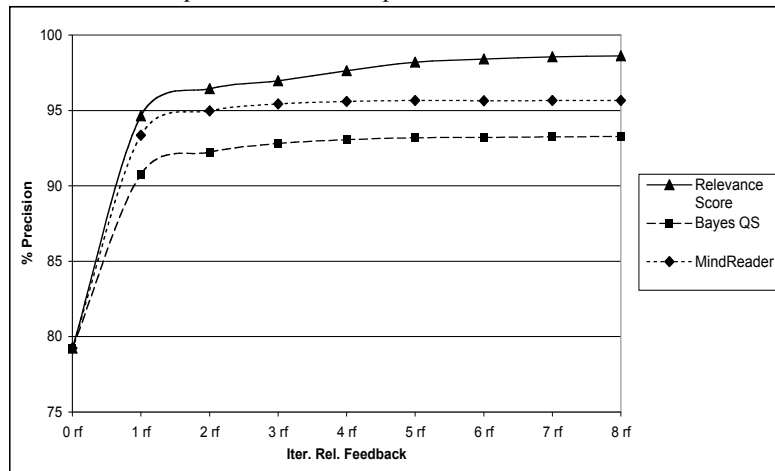

Figure 1: Retrieval Performances for the MIT database in terms of average percentage retrieval precision.

After the first feedback iteration (1rf in the graph), each relevance feedback mechanism is able to improve the average precision attained in the first retrieval by more than 10%, the proposed mechanism performing slightly better than MindReader. This is a desired behaviour as a user typically allows few iterations. However, if the user aims to better refine the search by additional feedback iteration, MindReader and Bayes QS are not able to exploit the additional information, as they provide no improvements after the second feedback iteration. On the other hand, the proposed mechanism provides further improvement in precision by increasing the number of iteration. These improvements are very small

because the first feedback already provides a high precision value, near to 95%.

## 4.2 Experiments with the UCI database

This database too can be considered of the "narrow domain" type as the images clearly belong to one of the seven data classes, and features have been extracted accordingly.

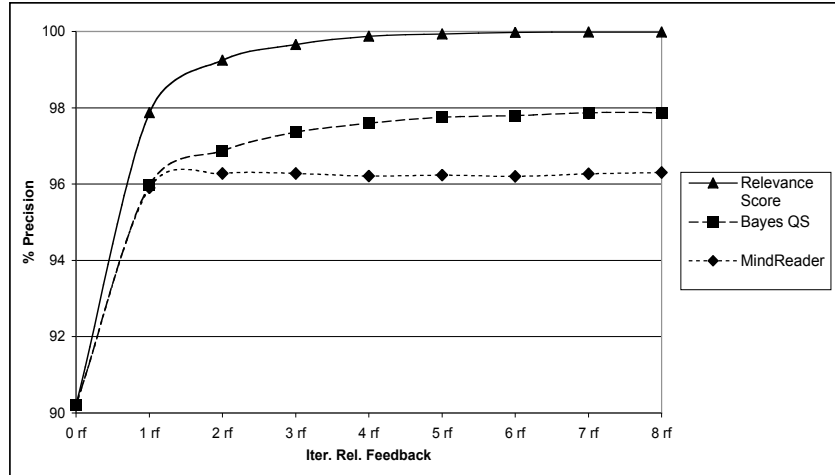

Figure 2: Retrieval Performances for the UCI data set in terms of average percentage retrieval precision.

Figure 2 show the performances attained on the UCI database. Retrieval precision is very high after the first extraction with no feedback. Nonetheless, each of the considered mechanism is able to exploit relevance feedback, Mindreader and Bayes QS providing a 6% improvement, while the proposed mechanism attains a 8% improvement. This example clearly shows the superiority of the proposed technique, as it attains a precision of 99% after the second iteration. Further iterations allow attaining a 100% precision. On the other hand, Bayes QS also exploits further feedback iteration attaining a precision of 98% after 7 iterations, while MindReader does not improve the precision attained after the first iteration. As the user typically allows very few feedback iterations, the proposed mechanism proved to be very suited for narrow domain databases as it allows attaining a precision close to 100%.

## 4.3 Experiments with the Corel database

Figures 3 and 4 show the performances attained on two feature sets extracted from the Corel database. This database is of the "broad domain" type as images represent a very large number of concepts, and the selected feature sets represent conceptual similarity between pairs of images only partly.

Reported results clearly show the superiority of the proposed mechanism. Let us note that the retrieval precision after the first $k$-nn search (0rf in the graphs) is quite small. This is a consequence of the difficulty of selecting a good feature space to represent conceptual similarity between pairs of images in a broad domain database. This difficulty is partially overcome by using MindReader or Bayes QS as they allow improving the retrieval precision by 10% to 15% according to the number of iteration allowed, and according to the selected feature space. Let us recall that both MindReader and Bayes QS perform a query movement in order to perform a k-nn

query on a more promising region of the feature space. On the other hand, the proposed mechanism based on ranking all the images of the database according to a relevance score, not only provided higher precision after the first feedback, but also allow to improve significantly the retrieval precision as the number of iteration is increased. As the initial precision is quite small, a user may have more willingness to perform further iterations as the proposed mechanism allows retrieving new relevant images.

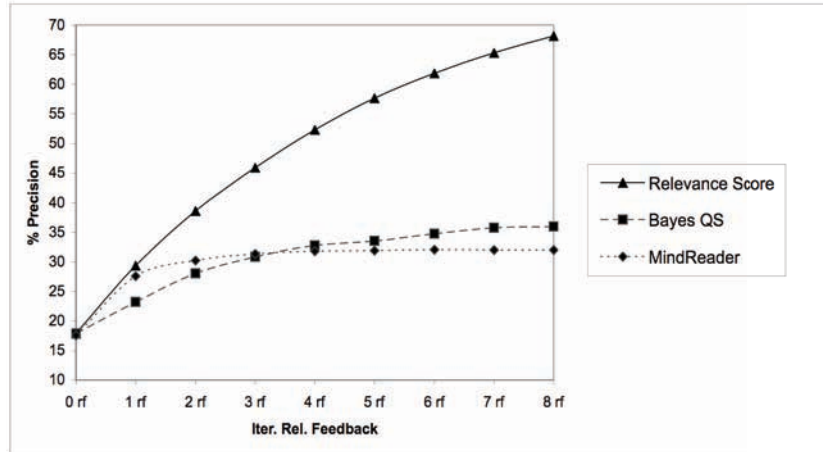

Figure 3: Retrieval Performances for the Corel data set (Color Moments feature set) in terms of average percentage retrieval precision

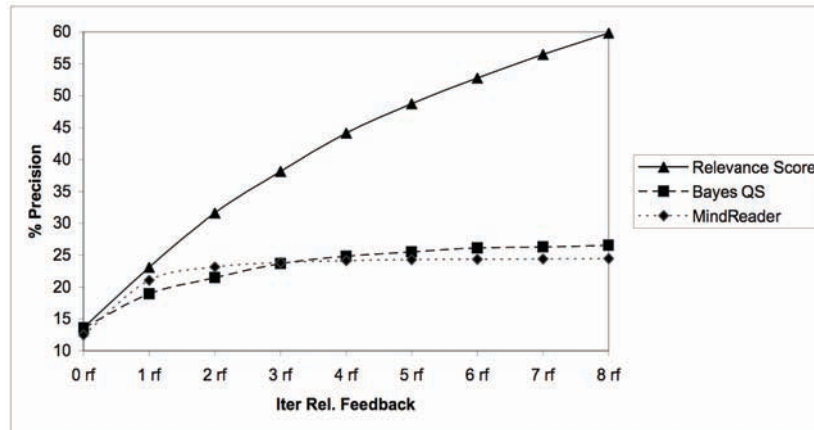

Figure 4: Retrieval Performances for the Corel data set (Co-occurrence Texture feature set) in terms of average percentage retrieval precision.

## 5 Conclusions

In this paper, we proposed a novel relevance feedback technique for content-based image retrieval. While the vast majority of relevance feedback mechanisms aims at modeling user's concept of relevance based on the available labeled samples, the proposed mechanism is based on ranking the images according to a relevance score depending on the dissimilarity from the nearest relevant and non-relevant images.

The rationale behind our choice is the same of case-based reasoning, instance-based learning, and nearest-neighbor pattern classification. These techniques provide good performances when the number of available *training* samples is too small to use statistical techniques. This is the case of relevance feedback in CBIR, where the use of classification models should require a suitable formulation in order to avoid so-called "small sample" problems.

Reported results clearly showed the superiority of the proposed mechanism especially when large databases made up of images related to many different concepts are searched. In addition, while many relevance feedback techniques require the tuning of some parameters, and exhibit high computational complexity, the proposed mechanism does not require any parameter tuning, and exhibit a low computational complexity, as a number of techniques are available to speed-up distance computations.

## References

[1] Smeulders A.W.M., Worring M., Santini S., Gupta A., Jain R.: Content-based image retrieval at the end of the early years. IEEE Trans. on Pattern Analysis and Machine Intelligence **22**(12) (2000) 1349-1380

[2] G. Salton and M.J. McGill, *Introduction to modern information retrieval,* New York, McGraw-Hill, 1988.

[3] Ishikawa Y., Subramanys R., Faloutsos C.: MindReader: Querying databases through multiple examples. In Proceedings. of the 24th VLDB Conference (1998) 433-438

[4] Santini S., Jain R.: Integrated browsing and querying for image databases. IEEE Multimedia **7**(3) (2000) 26-39

[5] Rui Y., Huang T.S.: Relevance Feedback Techniques in Image retrieval. In Lew M.S. (ed.): Principles of Visual Information Retrieval. Springer, London, (2001) 219-258

[6] Sclaroff S., La Cascia M., Sethi S., Taycher L.: Mix and Match Features in the ImageRover search engine. In Lew M.S. (ed.): Principles of Visual Information Retrieval. Springer-Verlag, London (2001) 219-258

[7] Giacinto G., Roli F.: Bayesian relevance feedback for content-based image retrieval. Pattern Recognition 37(7) (2004) 1499-1508

[8] Zhou X.S. and Huang T.S.: Relevance feedback in image retrieval: a comprehensive review, Multimedia Systems 8(6) (2003) 536-544

[9] Aha D.W., Kibler D., Albert M.K. Instance Based learning Algorithms. Machine Learning, 6, (1991) 37-66

[10] Althoff K-D. Case-Based Reasoning. In Chang S.K. (ed.) Handbook on Software Engineering and Knowledge Engineering, World Scientific (2001), 549-588.

[11] Bloch I. Information Combination Operators for Data Fusion: A Comparative Review with Classification. IEEE Trans. on System, Man and Cybernetics - Part A, 26(1) (1996) 52-67

[12] Duda R.O., Hart P.E., and Stork D.G.: Pattern Classification. John Wiley and Sons, Inc., New York, 2001

[13] Hastie T., Tibshrirani R., and Friedman J.: The Elements of Statistical Learning. Springer, New York, 2001

[14] Peng J., Bhanu B., Qing S., Probabilistic feature relevance learning for content-based image retrieval, Computer Vision and Image Understanding 75 (1999) 150-164.

[15] He J., Li M., Zhang H-J, Tong H., Zhang C, Mean Version Space: a New Active Learning Method for Content-Based Image Retrieval, Proc. of MIR 2004, New York, USA. (2004) 15-22.
